# Why MCA? Nonlinear sparse coding with spike-and-slab prior for neurally plausible image encoding

**Jacquelyn A. Shelton,   Philip Sterne,   Jörg Bornschein,   Abdul-Saboor Sheikh,**
Frankfurt Institute for Advanced Studies
Goethe-University Frankfurt, Germany
`{shelton, sterne, bornschein, sheikh}@fias.uni-frankfurt.de`

**Jörg Lücke**
Frankfurt Institute for Advanced Studies
Physics Dept., Goethe-University Frankfurt, Germany
`luecke@fias.uni-frankfurt.de`

## Abstract

Modelling natural images with sparse coding (SC) has faced two main challenges: flexibly representing varying pixel intensities and realistically representing low-level image components. This paper proposes a novel multiple-cause generative model of low-level image statistics that generalizes the standard SC model in two crucial points: (1) it uses a spike-and-slab prior distribution for a more realistic representation of component absence/intensity, and (2) the model uses the highly nonlinear combination rule of maximal causes analysis (MCA) instead of a linear combination. The major challenge is parameter optimization because a model with either (1) or (2) results in strongly multimodal posteriors. We show for the first time that a model combining both improvements can be trained efficiently while retaining the rich structure of the posteriors. We design an exact piecewise Gibbs sampling method and combine this with a variational method based on preselection of latent dimensions. This combined training scheme tackles both analytical and computational intractability and enables application of the model to a large number of observed and hidden dimensions. Applying the model to image patches we study the optimal encoding of images by simple cells in V1 and compare the model's predictions with *in vivo* neural recordings. In contrast to standard SC, we find that the optimal prior favors asymmetric and bimodal activity of simple cells. Testing our model for consistency we find that the average posterior is approximately equal to the prior. Furthermore, we find that the model predicts a high percentage of globular receptive fields alongside Gabor-like fields. Similarly high percentages are observed *in vivo*. Our results thus argue in favor of improvements of the standard sparse coding model for simple cells by using flexible priors and nonlinear combinations.

## 1   Introduction

Sparse Coding (SC) is one of the most popular algorithms for feature learning and has become a standard approach in Machine Learning, Computational Neuroscience, Computer Vision, and other related fields. It was first introduced as a model for the encoding of visual data in the primary visual cortex of mammals [1] and became the standard model to describe coding in simple cells. Following early recording studies [2] on simple cells, they were defined to be cells responding to localized, oriented and bandpass visual stimuli – sparse coding offered an optimal encoding explanation of such responses by assuming that visual components are (a) independent, (b) linearly superimposed,

and (c) mostly inactive, with only a small subset of active components for a given image patch. More formally, sparse coding assumes that each observation $\mathbf{y} = (y_1, \ldots, y_D)$ is associated with a (continuous or discrete) sparse latent variable $\mathbf{s} = (s_1, \ldots, s_H)$, where sparsity implies that most of the components $s_h$ in $\mathbf{s}$ are zero or close-to zero. Each data point is generated according to the data model

$$p(\mathbf{y} \,|\, \Theta) = \int_{\mathbf{s}} p(\mathbf{y} \,|\, \mathbf{s}, \Theta) \, p(\mathbf{s} \,|\, \Theta) \, d\mathbf{s} \tag{1}$$

with $\int_{\mathbf{s}}$ integrating (or summing) over all hidden states and $\Theta$ denoting the model parameters. Typically, $p(\mathbf{y} \,|\, \mathbf{s}, \Theta)$ is modelled as a Gaussian with a mean $\mu$ defined as $\mu = \sum_h s_h \mathbf{W}_h$, i.e. as a linear superposition of basis vectors $\mathbf{W}_h \in \mathbb{R}^D$. The most typical choice of prior over $p(\mathbf{s} \,|\, \Theta)$ is a Laplace distribution (which corresponds to L$_1$ regularization).

The sparse coding generative model has remained essentially the same since its introduction, with most work focusing on efficient inference of optimal model parameters $\Theta$ (e.g., [3, 4]), usually exploiting unimodality of the resulting posterior probabilities. The standard form of the model offers many mathematically convenient advantages, but the inherent assumptions may not be appropriate if the goal is to accurately model realistic images. First, it has been pointed out that visual components – such as edges – are either present or absent and this is poorly modelled with a Laplace prior because it lacks exact zeros. Recently, spike-and-slab distributions have been a favored alternative (e.g. [5, 6, 7, 8]) as they enable the modelling of visual component absence/presence (the spike) as well as the component's intensity distribution (the slab). Second, it has been pointed out that image components do not linearly superimpose to generate images, contrary to the standard sparse coding assumption. Alternatively, various nonlinear combinations of visual components have been investigated [9, 10, 11, 12]. Either modification (spike-and-slab prior or nonlinearities) leads to multimodal posteriors, making parameter optimization difficult. As a result these modifications have so far only been investigated separately. For linear sparse coding with a spike-and-slab prior the challenge for learning has been overcome by applying factored variational EM approaches [13, 5] or sampling [6]. Similarly, models with nonlinear superposition of components could be efficiently trained by applying a truncated variational EM approach [14, 12], but avoiding the analytical intractability introduced by using a continuous prior distribution.

In this work we propose a sparse coding model that for the first time combines both of these improvements – a *spike-and-slab distribution* and *nonlinear combination of components* – in order to form a more realistic model of images. We address the optimization of our model by using a combined approximate inference approach with preselection of latents (for truncated variational EM [14]) in combination with Gibbs sampling [15]. First, we show on artificial data that the method efficiently and accurately infers all model parameters, including data noise and sparsity. Second, using natural image patches we show the model yields results consistent with *in vivo* recordings and that the model passes a consistency check which standard SC does not. Third, we show our model performs on par with other models on the functional benchmark tasks of denoising and inpainting.

## 2  The Generative Model: Nonlinear Spike-and-Slab Sparse Coding

We formulate the multi-causal data generation process as the probabilistic generative model:

$$p(y_d \,|\, \mathbf{s}, \Theta) = \mathcal{N}(y_d; \, \max_h\{s_h W_{dh}\}, \sigma^2), \tag{2}$$

where $\max_h$ considers all $H$ latent components and takes the $h$ yielding the maximum value for $s_h W_{dh}$, and where $s_h$ has a spike-and-slab distribution given by $s_h = b_h z_h$ and parameterized by:

$$p(b_h \,|\, \Theta) = \mathcal{B}(b_h; \pi) = \pi^{b_h} \, (1 - \pi)^{1 - b_h} \tag{3}$$

$$p(z_h \,|\, \Theta) = \mathcal{N}(z_h; \, \mu_{\mathrm{pr}}, \sigma_{\mathrm{pr}}^2), \tag{4}$$

in this notation the spike is defined in Eq. 3 (parameterized by $\pi$) and the slab is defined in Eq. 4 (parameterized by $\mu_{\mathrm{pr}}$ and $\sigma_{\mathrm{pr}}$). The observation noise has a single parameter; $\sigma$. The columns of the matrix $W = (W_{dh})$ are the generative fields, $\mathbf{W}_h$, one associated with each latent variable $s_h$. We denote the set of all parameters with $\Theta$. We will be interested in working with the posterior over the latents given by

$$p(\mathbf{s}|\mathbf{y}, \theta) = \frac{p(\mathbf{y}|\mathbf{s}, \theta) \, p(\mathbf{s}|\theta)}{\int_{s'} p(\mathbf{y}|\mathbf{s}', \theta) \, p(\mathbf{s}'|\theta) \, d\mathbf{s}'}. \tag{5}$$

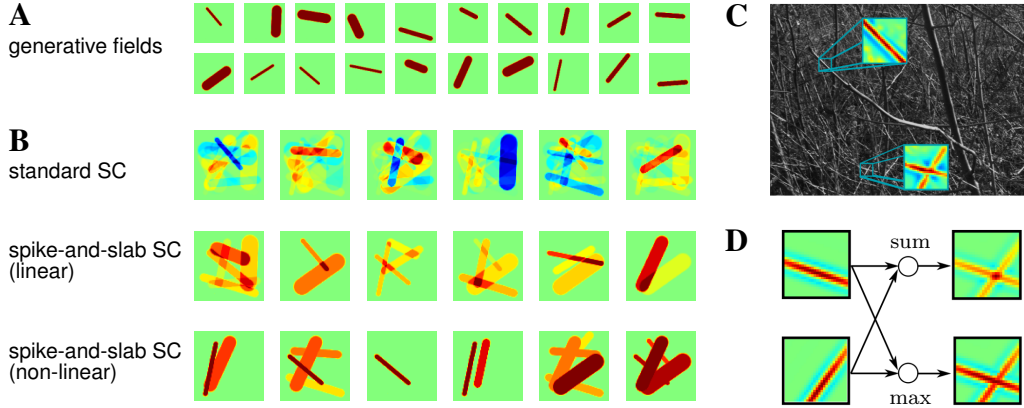

Figure 1: Generation according to different sparse coding generative models using the same generative fields. **A** 20 generative fields in the form of random straight lines. **B** Examples of patches generated according to three generative models all using the fields in **A**. Top row: standard linear sparse coding with Laplace prior. Middle row: linear sparse coding with spike-and-slab prior. Bottom row: spike-and-slab sparse coding with max superposition. The latter two use the same prior parameterization (with positive slab). Generated patches are individually scaled to fill the color space (with zero fixed to green). **C** A natural image with two patches highlighted (magnifications show their preprocessed from). **D** Linear and nonlinear superposition of two single components for comparison with the actual superposition in **C**.

As in standard sparse coding, the model assumes independent latents and given the latent variables, the observations are distributed according to a Gaussian distribution. Unlike standard sparse coding, the latent variables are not distributed according to a Laplace prior and the generative fields (or *basis functions*) are not combined linearly. Fig. 1 illustrates the model differences between a Laplace prior and a spike-and-slab prior and the differences between linear and nonlinear superposition. As can be observed, standard sparse coding results in strong interference when basis functions overlap. For spike-and-slab sparse coding most components are exactly zero but interference between them remains strong because of their linear superposition. Combining a spike-and-slab prior with non-linear composition allows minimal interference between the bases and ensures that latents can be exactly zero, which creates very multimodal posteriors since data must be explained by either one cause or another. For comparison, the combination of two real image components is highlighted in Fig. 1**C** (lower patch). Linear and nonlinear superposition of two basis functions resembling single components is shown in Fig. 1**D**. This suggests that superposition defined by max represents a better model of occluding components (compare [11, 12]).

In this paper we use expectation maximization (EM) to estimate the model parameters $\Theta$, and we use sampling after latent preselection [15] to represent the posterior distribution over the latent space. Optimization in the EM framework entails setting the free-energy to zero and solving for the model parameters (M-step equations) (e.g., [16]). As an example we obtain the following formula for the estimate of image noise:

$$\hat{\sigma}^2 = \frac{1}{NDK} \sum_n \sum_d \sum_k \left( \max_h \left\{ W_{hd} s_h^k \right\} - y_d^{(n)} \right)^2 , \qquad (6)$$

where we average over all $N$ observed data points, $D$ observed dimensions, and $K$ Gibbs samples. However, this notation is rather unwieldy for a simple underlying idea. As such we will use the following notation:

$$\hat{\sigma}^2 = \left\langle W_{dh} s_h - y_d^{(n)} \right\rangle^* , \qquad (7)$$

where we maximize for $h$ and average over $n$ and $d$. That is, we denote the expectation values $\langle . \rangle^*$ to mean the following:

$$\langle f(s) \rangle^* = \sum_n \frac{\int_s p(\mathbf{s}|\mathbf{y}^{(n)}, \Theta) \, f(\mathbf{s}) \, \delta(\text{h is max}) \, d\mathbf{s}}{\int_s p(\mathbf{s}|\mathbf{y}^{(n)}, \Theta) \, \delta(\text{h is max}) \, d\mathbf{s}} , \qquad (8)$$

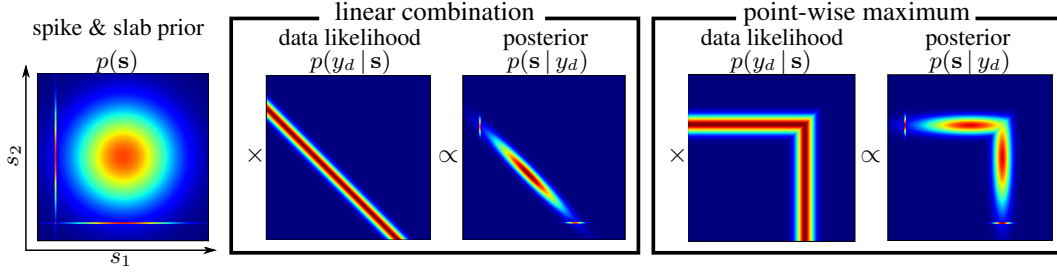

Figure 2: Illustration of a H=2-dimensional spike-and-slab prior over latents and the multimodal posterior distribution induced by this prior for both linear and nonlinear data likelihoods.

where $\delta$ is the indicator function denoting the domain to integrate over, namely where $h$ is the maximum. This allows a condensed expression of the rest of the update equations:

$$\hat{W}_{hd} = \frac{\langle s_h y_d \rangle^*}{\langle s_d^2 \rangle^*}, \qquad\qquad \hat{\pi} = \langle \delta(s) \rangle, \tag{9}$$

$$\hat{\mu}_{\mathrm{pr}} = \langle s_h \rangle^*, \qquad\qquad \hat{\sigma}_{\mathrm{pr}}^2 = \langle (s_h - \hat{\mu}_{\mathrm{pr}})^2 \rangle^* \tag{10}$$

where we observe that in order to optimize the parameters we need to calculate several expectation values with respect to the posterior distribution. As discussed however, the posterior distribution of a model with a spike-and-slab prior in both the linear and nonlinear cases is strongly multimodal and such posteriors are difficult to infer and represent. This is illustrated in Fig. 2. Calculating expectations of this posterior is analytically intractable, thus we use Gibbs sampling to approximate the expectations.

## 3 Inference: Exact Gibbs Sampling with Preselection of Latents

In order to efficiently handle the intractabilities posed by our model and the complex posterior (multimodal, high dimensional) illustrated in Fig. 2, we take a combined approximate inference approach. Specifically we do exact Gibbs sampling from the posterior after we have preselected the most relevant set of latent states using a truncated variational form of EM. Preselection is not strictly necessary, but significantly helps with the computational intractability faced in high dimensions. As such, we will first descibe the sampling step and preselection only later.

**Gibbs Sampling.** Our main technical contribution towards efficient inference in this model is an *exact Gibbs sampler for the multimodal posterior*. Previous work has used Gibbs sampling in combination with spike-and-slab models [17], and for increased efficiency in sparse Bayesian inference [18]. Our aim is to construct a Markov chain with the target density given by the conditional posterior distribution:

$$p(s_h | \mathbf{s}_{H \setminus h}, \mathbf{y}, \theta) \ \propto \ p(s_h | \theta) \prod_{d=1}^{D} p(y_d | s_h, \mathbf{s}_{H \setminus h}, \theta). \tag{11}$$

We see from Eq. 11 that the distribution factorizes into $D + 1$ factors: a *single factor* for the *prior* and $D$ *factors* for *each likelihood*. For the point-wise maximum nonlinear case we are considering, the likelihood of a single $y_d$ is a piecewise function defined as follows:

$$p(y_d | s_h, \mathbf{s}_{H \setminus h}, \theta) = \mathcal{N}(y_d; \max_{h'} \{W_{dh'} s_{h'}\}, \sigma^2) \tag{12}$$

$$= \begin{cases} \underbrace{\mathcal{N}(y_d; \max_{h' \setminus h} \{W_{dh'} s_{h'}\}, \sigma^2)}_{\text{constant}} = \exp(l_d(s_h)) & \text{if } s_h < P_d \\ \mathcal{N}(y_d; W_{dh} s_h, \sigma^2) = \exp(r_d(s_h)) & \text{if } s_h \geq P_d, \end{cases} \tag{13}$$

where the transition point is defined as the point where $s_h W_{dh}$ becomes the maximal cause:

$$P_d = \max_{h' \setminus h} \{W_{dh'} s_{h'}\} / W_{dh}. \tag{14}$$

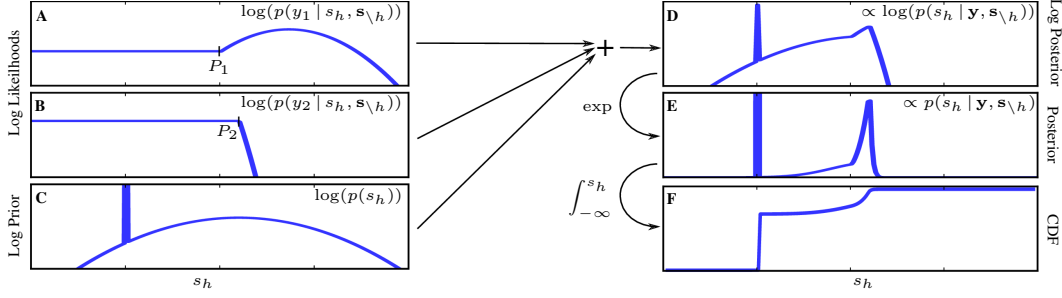

Figure 3: Illustration of the Gibbs sampler for an MCA-induced posterior. Left column: three contributing factors for the posterior $\propto p(s_h \,|\, s_{\backslash h}, \mathbf{y}, \Theta)$ in logspace. **A** and **B**: Log likelihood functions each defined by a transition point $P_d$ and left and right pieces $r_d(s_h)$ and $l_d(s_h)$. **C** Log prior, which consists of an overall gaussian and the Dirac-peak at $s_h = 0$. **D** Log posterior, the sum of functions **A**, **B**, and **C** consists of $D+1$ pieces plus the Dirac-peak at $s_h = 0$. **E** Exponentiation of the **D** log posterior. **F** CDF for $s_h$ from which we do inverse transform sampling.

We refer to the two pieces of $y_d$ in Eq. 13 as the *left piece* of the function when $s_h < P_d$ and *right piece* when $s_h \geq P_d$. The left is constant w.r.t. $s_h$ because the data is explained by another cause when $s_h < P_d$, and the right is a truncated Gaussian when considered a PDF of $s_h$ (see Fig. 3**A**).

We take the logarithm of $p(y_d|s_h, \mathbf{s}_{H\backslash h}, \theta)$, which transforms the equation into a left-piece constant and right-piece quadratic function that we can easily sum together. The sum of these $D$ functions results in one function with $D+1$ segments $m_i(s_h)$, with transition points $P_d$ given by the individual functions. We first sort the functions according to their transition points, denoted here by $\delta = \mathrm{argsort}_d(P_d)$, such that we can efficiently calculate the summation over these functions:

$$m_i(s_h) = \sum_{j=1}^{i-1} r_{\delta(j)}(s_h) + \sum_{u=i}^{D} l_{\delta(u)}(s_h) \qquad \text{for} \quad 1 \leq i \leq D+1, \qquad (15)$$

where the left and right pieces are referred to as $l_i(s_h)$ and $r_i(s_h)$ (as in Eq. 13), respectively. Since all pieces $l_i(s_h)$ and $r_i(s_h)$ are polynomials of 2nd degree, the result is still a 2nd degree polynomial. We incoorporate the prior in two steps. The *Gaussian slab of the prior* is taken into account by adding its 2nd degree polynomial to all the pieces $m_i(s_h)$, which also ensures that every piece is a Gaussian.

To construct the piecewise cumulative distribution function (CDF), we relate each segment $m_i(s_h)$ to the Gaussian $\propto \exp(m_i(s_h))$ it defines. Next, the *Bernoulli component of the prior* is accounted for by introducing the appropriate step into the CDF at $s_h = 0$ (see Fig. 3**F**). Once the CDF is constructed, we simulate each $s_h$ from the exact conditional distribution $(s_h \sim p(s_h|\mathbf{s}_{\backslash h} = \mathbf{s}_{\backslash h}, \mathbf{y}, \theta))$ by inverse transform sampling. Fig. 3 illustrates the entire process.

**Preselection.** To reduce the computational load of inference in our model, we can optionally preselect the most relevant latent variables before doing Gibbs sampling. This can be formulated as a variational approximation to exact inference [14] where the posterior distribution $p(\mathbf{s} \,|\, \mathbf{y}^{(n)}, \Theta)$ is approximated by the distribution $q_n(\mathbf{s}; \Theta)$ which only has support on a subset $\mathcal{K}_n$ of the latent state space:

$$p(\mathbf{s} \,|\, \mathbf{y}^{(n)}, \Theta) \approx q_n(\mathbf{s}; \Theta) = \frac{p(\mathbf{s} \,|\, \mathbf{y}^{(n)}, \Theta)}{\displaystyle\int_{\mathbf{s}' \in \mathcal{K}_n} p(\mathbf{s}' \,|\, \mathbf{y}^{(n)}, \Theta)} \, \delta(\mathbf{s} \in \mathcal{K}_n) \qquad (16)$$

where $\delta(\mathbf{s} \in \mathcal{K}_n) = 1$ if $\mathbf{s} \in \mathcal{K}_n$ and zero otherwise. The subsets $\mathcal{K}_n$ are chosen in a data-driven way using a deterministic *selection function*, they vary per data point $\mathbf{y}^{(n)}$, and should contain most of the probability mass $p(\mathbf{s} \,|\, \mathbf{y})$ while also being significantly smaller than the entire latent space. Using such subsets $\mathcal{K}_n$, Eqn. 16 results in good approximations to the posteriors. We define $\mathcal{K}_n$ as $\mathcal{K}_n = \{\mathbf{s} \,|\, \text{for all } h \notin \mathcal{I} : \ s_h = 0\}$ where $\mathcal{I}$ contains the indices of the latents estimated to be most relevant for $\mathbf{y}^{(n)}$. To obtain these latent indices we use a selection function of the form:

$$\mathcal{S}_h(\mathbf{y}^{(n)}) \quad = \quad \left|\mathbf{W}_h - \mathbf{y}^{(n)}\right|_2^2 \Big/ \left|\mathbf{W}_h\right|_2 \qquad (17)$$

to select the $H' < H$ highest scoring latents for $\mathcal{I}$. This boils down to selecting the $H'$ dictionary elements that are most similar to each datapoint, hence being most likely to have generated the datapoint. We then sample from this reduced latent set.

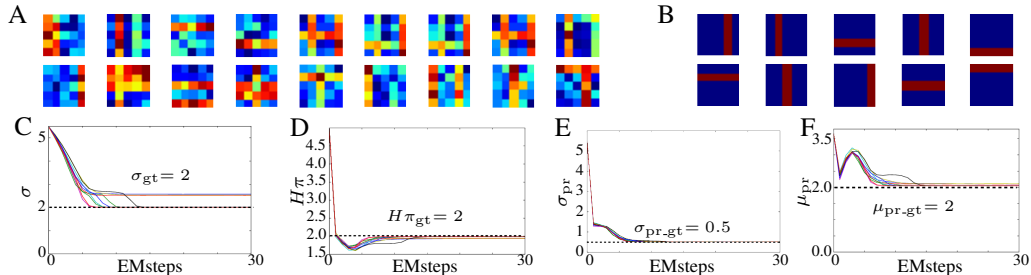

Figure 4: Results of 10 experimental runs with 30 EM iterations on the same artificial ground-truth data generated according to the model. We accurately recover ground-truth parameters which are plotted with dotted lines. **A** Random selection of data $\mathbf{y}^{(n)}$, **B** The set of learned generative fields $\mathbf{W}_h$, **C** Data noise $\sigma$, **D** Sparsity $H \times \pi$, **E** Prior standard dev. $\sigma_{\mathrm{pr}}$, **F** Prior mean $\mu_{\mathrm{pr}}$.

## 4    Experiments

We first investigate the performance of our algorithm on ground-truth artificial data. Second, we apply our model to natural image patches and compare with *in vivo* recording from various sources. Third, we investigate the applicability of our algorithm on functional benchmark tasks. All experiments were performed using a parallel implementation of the EM algorithm [19]. Small scale experiments were run on a single multicore machine, while larger scale experiments were typically run on a cluster with 320 CPU cores in parallel.

For all experiments described, $1/3$ of the samples drawn are used for burn-in, and $2/3$ are the samples used for computations. We initialize our parameters by setting the $\sigma_{\mathrm{pr}}$ and $\sigma$ equal to the standard deviation observed in the data, the prior mean $\mu_{\mathrm{pr}}$ is initialized to the observed data mean. $W$ is initialized at the observed data mean with additive Gaussian noise of the $\sigma$ observed in the data, but we enforce the constraint that $\langle W_{dh} \rangle = 1$ such that $\forall_{h \in H} \sum_{d=1}^{D} W_{dh} = D$, or that each $W_{dh}$ is approximately equal to one.

**Artificial Data.** The goal of the first set of experiments is to verify that our model and inference method produce an algorithm that can (1) recover ground-truth parameters from data that is generated according to the model and (2) that it reliably converges to locally optimal solutions. We generate ground-truth data with $N = 2,000$ consisting of $D = 5 \times 5 = 25$ observed and $H = 10$ hidden dimensions according to our model: $N$ images with overlapping 'bars' of varying intensities and with Gaussian observation noise of variance $\sigma_{gt} = 2$ (Fig. 4**A**). On average, each data point contains two bars, $\pi = \frac{2}{H}$. Results (Fig. 4**B**-**F**) show that our algorithm converges to globally optimal solutions and recovers the generating ground-truth parameters. Here we drew 30 samples from the posterior and set $H' = H$, but investigation of a range of sample number and $H'$ values yields the same results, suggesting that our approximation parameters do not have an effect on our results (see Supp. Material for more experiments on this dataset).

**Natural Image Patches.** We applied our model to $N = 50,000$ image patches of $16 \times 16$ pixels. The patches were extracted from the Van Hateren natural image database [20] and subsequently preprocessed using pseudo-whitening [1]. We split the image patches into a positive and negative channel to ensure $y_d \geq 0$: each image patch $\tilde{y}$ of size $\tilde{D} = 16 \times 16$ is converted into a datapoint of size $D = 2\tilde{D}$ by assigning $y_d = [\tilde{y}_d]^+$ and $y_{\tilde{D}+d} = [-\tilde{y}_d]^+$, where $[x]^+ = x$ for $x > 0$ and $[x]^+ = 0$ otherwise. This can be motivated by the transfer of visual information by center-on and center-off cells of the mammalian lateral geniculate nucleus (LGN). In a final step, as a form of local contrast normalization, we scaled each image patch so that $0 \leq y_d \leq 10$.

After 50 EM iterations with 100 samples per datapoint the model parameters had converged and the learned dictionary elements $\mathbf{W}_h$ represent a variety of Gabor-Wavelet and Difference of Gaussians (DoG) like shapes (see Fig. 5**A**). We observe a mean activation of $\mu_{pr} = 0.47$, with standard deviation $\sigma_{pr} = 0.13$, i.e., we infer a strongly bimodal prior (Fig. 5**D**). The final sparseness was $\pi H = 6.2$, which means that an average of roughly six latent variables were active in every image patch. The inferred observation noise was $\sigma = 1.4$. To quantitatively interpret the learned fields, we

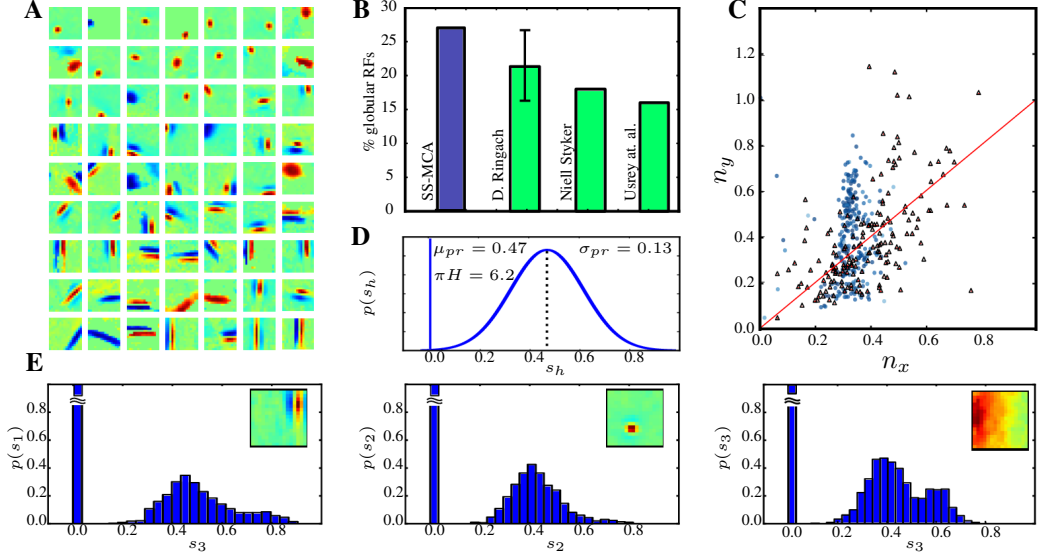

Figure 5: Results after training our model on $N = 50,000$ image patches of size $16 \times 16$ using H=500 latent units. **A** Selection of inferred dictionary elements $\mathbf{W}_h$. The full set of elements is shown in the supplementary material. **B** After fitting with Gabor wavelets and DoG's, 135 fields (27%) are classified as being globular. The fraction of globular fields measured *in vivo* are shown for comparison. **C** $n_x/n_y$ Gabor statistics plot of estimated receptive fields (blue circles, see Supp. D Fig.**D**) overlaid with the distribution reported by Ringach (*in vivo* Macaque, red triangles). We intentionally exclude fields best fit by DoG's, removing the typical cluster observed at 0-0 (see Supp. D). **D** Visualization of the prior inferred by our model: On average $\pi H = 6.2$ dictionary elements are active per datapoint. **E** Histogram of the actual activation of three selected dictionary elements: A Gabor wavelet, a DoG and a field encoding low-frequency input. The bimodal pattern closely resembles the prior activation inferred in **D**.

perform reverse correlation on the learned generative fields and fit the resulting estimated receptive fields with Gabor wavelets and DoGs (see Supp. D for details). Next, we classify the fields as either orientation-sensitive Gabor wavelets or 'globular' fields best matched by DoGs. In Fig. 5**C**) we then plot only the fields classified as Gabors, leaving out all DoG fields.

Notably, the proportion of globular fields predicted by the model (Fig. 5**B**) is similarly high as those found in different species [21, 22, 23] (see next section for a discussion). Fig. 5**D-E** compares the optimal prior distribution with the average posterior distribution for several latent variables (with their associated generative fields shown in insets). It is a necessary condition of the correct model of the data that the posterior averaged over the datapoints $\mathbf{y}^{(n)}$ matches the prior, since the following holds (compare, e.g., [24]):

$$\lim_{N \to \infty} \tfrac{1}{N} \textstyle\sum_n p(\mathbf{s} \,|\, \mathbf{y}^{(n)}, \Theta) = p(\mathbf{s} \,|\, \Theta). \tag{18}$$

Our model satisfies this condition; the average posterior over these fields closely resembles the optimal prior, which is a test standard sparse coding fails (see [17] for a discussion).

**Functional Tasks.** We also apply our model to the task of image inpainting and image denoising. Given that we propose our model to be able to realistically model low-level image statistics, we expect it to perform well on these tasks. Results show that our algorithm performs on par with the latest benchmarks obtained by other algorithms. See Supp. Material for details and examples.

## 5  Discussion

In this work, we defined and studied a sparse coding model that, for the first time, combines a spike-and-slab prior with a nonlinear combination of dictionary elements. To address the optimization of our model, we designed an exact piecewise Gibbs sampling method combined with a variational method based on preselection of latent dimensions. This combined training scheme tackles both

analytical and computational intractability and enables application of the model to a large number of observed and hidden dimensions. The learning algorithm derived for the model enables the efficient inference of all model parameters including sparsity and prior parameters.

The spike-and-slab prior used in this study can parameterize prior distributions which are symmetric and unimodal (spike on top of the Guassian) as well as strongly bimodal distributions with the Gaussian mean being significantly different from zero. However, inferring the correct prior distribution requires sophisticated inference and learning schemes. Standard sparse coding with MAP-based approximation only optimizes the basis functions [25, 4]. Namely, the prior shape remains fixed except for its weighting factor (the regularization parameter) which is typically only inferred indirectly (if at all) using cross-validation. Very few sparse coding approaches infer prior parameters directly. One example is an approach using a mixture-of-Gaussians (MoG) prior [17] which applies Gibbs sampling for inference. The MoG prior can model multimodality but in numerical experiments on image patches the mixture components were observed to converge to a monomodal prior – which may be caused by the assumed linear superposition or by the Gibbs sampler not mixing sufficiently. When the MoG prior was fixed to be trimodal, no instructive generative fields were observed [17]. Another example of sparse coding with prior inference is a more recent approach which uses a parameterized student-t distribution as prior and applies sampling to infer the sparsity [26]. A student-t distribution cannot model multimodality, however. The work in [27] uses a trimodal prior for image patches but shape and sparsity remain fixed, i.e. the study does not answer how optimal such a prior may be. In contrast, we have shown in this study that the prior shape and sparsity level can be inferred from image data. The resulting prior is strongly bimodal and control experiments confirm a high consistency of the prior with the average posterior (Fig. 5**D-E**). Standard sparse coding approaches typically fail in such controls which may be taken as early evidence for bimodal or multimodal priors being more optimal (see [17]).

Together with a bimodal prior, our model infers Gabor and difference-of-Gaussian (DoG) functions as the optimal basis functions for the used image patches. While Gabors are the standard outcome of sparse coding, DoGs have not been predicted by sparse coding until very recently. Indeed, DoG or 'globular' fields were identified as the main qualitative difference between experimental measurements of V1 simple cells and theoretical predictions [21]. A number of studies have since shown that globular fields can emerge in applications of computational models to image patches [28, 27, 29, 30, 31, 12, 32]. One study [29] has shown that globular fields can be obtained with standard sparse coding by choosing specific values for overcompleteness and sparsity (i.e. prior shape and sparsity are not inferred from data). The studies [27, 31, 32] assume a restricted set of values for latent variables and yield relatively high proportion of globular fields suggesting that the emergence of globular fields is due to hard constraints on the latents. On the other hand, the studies [28, 30, 12] suggest that globular fields are a consequence of occlusion nonlinearities. Our study argues in favor of the occlusion interpretation for the emergence of globular fields because the model studied here shows that high percentages of globular fields emerge with a prior that is (a) inferred from data and (b) allows for a continuous distribution of latent values.

In summary, the main results obtained by applying the novel model to preprocessed images are: (1) the observation that a bimodal prior is preferred over a unimodal one for optimal image coding, and (2) that high percentages of globular fields are predicted. The sparse bimodal prior is consistent with sparse and positive neural activtiy for the encoding of image components in V1, and the high percentage of globular fields is consistent with recent *in vivo* recordings of simple cells. Our model therefore links improvements on optimal image encoding to a high consistency with neural data.

**Acknowledgements.** We acknowledge support by the German Research Foundation (DFG) in the project LU 1196/4-2, by the German Federal Ministry of Education and Research (BMBF), project 01GQ0840, and by the LOEWE Neuronale Koordination Forschungsschwerpunkt Frankfurt (NeFF). Furthermore, we acknowledge support by the Frankfurt Center for Scientific Computing (CSC).

# References

[1] B. Olshausen and D. Field. Emergence of simple-cell receptive field properties by learning a sparse code for natural images. *Nature*, 381:607–9, 1996.

[2] D. H. Hubel and T. N. Wiesel. Receptive fields of single neurones in the cat's striate cortex. *The Journal of Physiology*, 1959.

[3] M. Seeger. Bayesian inference and optimal design for the sparse linear model. *Journal of Machine Learning Research*, 9:759–813, June 2008.

[4] H. Lee, A. Battle, R. Raina, and A. Ng. Efficient sparse coding algorithms. In *Advances in Neural Information Processing Systems*, volume 20, pages 801–08, 2007.

[5] M. Titsias and M. Lázaro-Gredilla. Spike and slab variational inference for multi-task and multiple kernel learning. In *Advances in Neural Information Processing Systems*, 2011.

[6] S. Mohamed, K. Heller, and Z. Ghahramani. Evaluating Bayesian and L1 approaches for sparse unsupervised learning. In *ICML*, 2012.

[7] I. Goodfellow, A. Courville, and Y. Bengio. Large-scale feature learning with spike-and-slab sparse coding. In *ICML*, 2012.

[8] Jörg Lücke and Abdul-Saboor Sheikh. Closed-form EM for sparse coding and its application to source separation. In *LVA/ICA*, LNCS, pages 213–221. Springer, 2012.

[9] E. Saund. A multiple cause mixture model for unsupervised learning. *Neural Computation*, 1995.

[10] P. Dayan and R. S. Zemel. Competition and multiple cause models. *Neural Computation*, 1995.

[11] J. Lücke and M. Sahani. Maximal causes for non-linear component extraction. *Journal of Machine Learning Research*, 9:1227–67, 2008.

[12] G. Puertas, J. Bornschein, and J. Lücke. The maximal causes of natural scenes are edge filters. In *Advances in Neural Information Processing Systems*, volume 23, pages 1939–47. 2010.

[13] I. Goodfellow, A. Courville, and Y. Bengio. Spike-and-slab sparse coding for unsupervised feature discovery. In *NIPS Workshop on Challenges in Learning Hierarchical Models*. 2011.

[14] Jörg Lücke and Julian Eggert. Expectation truncation and the benefits of preselection in training generative models. *Journal of Machine Learning Research*, 11:2855–900, 2010.

[15] J. Shelton, J. Bornschein, A.-S. Sheikh, P. Berkes, and J. Lücke. Select and sample - a model of efficient neural inference and learning. *Advances in Neural Information Processing Systems*, 24, 2011.

[16] R. Neal and G. Hinton. A view of the EM algorithm that justifies incremental, sparse, and other variants. In M. I. Jordan, editor, *Learning in Graphical Models*. Kluwer, 1998.

[17] B. Olshausen and K. Millman. Learning sparse codes with a mixture-of-Gaussians prior. *Advances in Neural Information Processing Systems*, 12:841–847, 2000.

[18] X. Tan, J. Li, and P. Stoica. Efficient sparse Bayesian learning via Gibbs sampling. In *ICASSP*, pages 3634–3637, 2010.

[19] J. Bornschein, Z. Dai, and J. Lücke. Approximate EM learning on large computer clusters. In *NIPS Workshop: LCCC*. 2010.

[20] J. H. van Hateren and A. van der Schaaf. Independent component filters of natural images compared with simple cells in primary visual cortex. *Proceedings of the Royal Society of London B*, 265:359–66, 1998.

[21] D. Ringach. Spatial structure and symmetry of simple-cell receptive fields in macaque primary visual cortex. *Journal of Neurophysiology*, 88:455–63, 2002.

[22] W. M. Usrey, M. P. Sceniak, and B. Chapman. Receptive Fields and Response Properties of Neurons in Layer 4 of Ferret Visual Cortex. *Journal of Neurophysiology*, 89:1003–1015, 2003.

[23] C. Niell and M. Stryker. Highly Selective Receptive Fields in Mouse Visual Cortex. *The Journal of Neuroscience*, 28(30):7520–7536, 2008.

[24] P. Berkes, G. Orban, M. Lengyel, and J. Fiser. Spontaneous Cortical Activity Reveals Hallmarks of an Optimal Internal Model of the Environment. *Science*, 331(6013):83–87, January 2011.

[25] B. Olshausen and D. Field. Sparse coding with an overcomplete basis set: A strategy employed by V1? *Vision Research*, 37(23):3311–3325, December 1997.

[26] P. Berkes, R. Turner, and M. Sahani. On sparsity and overcompleteness in image models. *Advances in Neural Information Processing Systems*, 21, 2008.

[27] M. Rehn and F. Sommer. A network that uses few active neurones to code visual input predicts the diverse shapes of cortical receptive fields. *Journal of Computational Neuroscience*, 22(2):135–46, 2007.

[28] J. Lücke. A dynamical model for receptive field self-organization in V1 cortical columns. In *Proc. International Conference on Artificial Neural Networks*, LNCS 4669, pages 389 – 398. Springer, 2007.

[29] B. A. Olshausen, C. Cadieu, and D.K. Warland. Learning real and complex overcomplete representations from the statistics of natural images. *Proc. SPIE*, (7446), 2009.

[30] J. Lücke. Receptive field self-organization in a model of the fine-structure in V1 cortical columns. *Neural Computation*, 21(10):2805–45, 2009.

[31] M. Henniges, G. Puertas, J. Bornschein, J. Eggert, and J. Lücke. Binary Sparse Coding. In *Proceedings LVA/ICA*, LNCS 6365, pages 450–57. Springer, 2010.

[32] J. Zylberberg, J. Murphy, and M. Deweese. A Sparse Coding Model with Synaptically Local Plasticity and Spiking Neurons Can Account for the Diverse Shapes of V1 Simple Cell Receptive Fields. *PLoS Computational Biology*, 7(10):e1002250, 2011.

[33] M. Zhou, H. Chen, J. Paisley, L. Ren, G. Sapiro, and L. Carin. Non-parametric Bayesian dictionary learning for sparse image representations 1. In *NIPS Workshop*. 2009.

